# Bayesian Estimation of Time-Frequency Coefficients for Audio Signal Enhancement

**Patrick J. Wolfe**[*]
Department of Engineering
University of Cambridge
Cambridge CB2 1PZ, UK
pjw47@eng.cam.ac.uk

**Simon J. Godsill**
Department of Engineering
University of Cambridge
Cambridge CB2 1PZ, UK
sjg@eng.cam.ac.uk

## Abstract

The Bayesian paradigm provides a natural and effective means of exploiting prior knowledge concerning the time-frequency structure of sound signals such as speech and music—something which has often been overlooked in traditional audio signal processing approaches. Here, after constructing a Bayesian model and prior distributions capable of taking into account the time-frequency characteristics of typical audio waveforms, we apply Markov chain Monte Carlo methods in order to sample from the resultant posterior distribution of interest. We present speech enhancement results which compare favourably in objective terms with standard time-varying filtering techniques (and in several cases yield superior performance, both objectively and subjectively); moreover, in contrast to such methods, our results are obtained without an assumption of prior knowledge of the noise power.

## 1 Introduction

Natural sounds can be meaningfully represented as a superposition of translated and frequency-modulated versions of simple functions (atoms). As a result, so-called time-frequency representations are ubiquitous in audio signal processing. The focus of this paper is on signal enhancement via a regression in which time-frequency atoms form the regressors. This choice is motivated by the notion that an atomic time-frequency decomposition is the most natural way to split an audio waveform into its constituent parts—such as note attacks and steady pitches for music, voiced and unvoiced speech, and so on. Moreover, these features, along with prior knowledge concerning their generative mechanisms, are most easily described jointly in time and frequency through the use of *Gabor frames*.

### 1.1 Gabor Frames

We begin by briefly reviewing the concept of Gabor systems; detailed results and proofs may be found in, for example, [1]. Consider a function $g$ whose time-frequency support

---

[*]Audio examples described in this paper, as well as Matlab code allowing for their reproduction, may be found at the author's web page: http://www-sigproc.eng.cam.ac.uk/~pjw47.

is centred about the origin, and let $g_{m,n}$ denote a time-shifted (translation by $na$) and frequency-shifted (modulation by $mb$) version thereof; such a collection of shifts defines a sampling grid over the time-frequency plane. Then (roughly speaking) if $g$ is reasonably well-behaved and the lattice $a\mathbb{Z} \times b\mathbb{Z}$ is sufficiently dense, the *Gabor system* $(g, a, b)$ provides a (possibly non-orthogonal, or even redundant) series expansion of any function in a Hilbert space, and is thus said to generate a frame.

More formally, a *Gabor frame* $(g_{m,n})$ is a dictionary of time-frequency shifted versions of a single basic window function $g$, having the additional property that there exist constants $A, B > 0$ (frame bounds) such that

$$A||f||^2 \leq \sum_{m,n \in \mathbb{Z}} |\langle f, g_{m,n} \rangle|^2 \leq B||f||^2 \ \forall f \in \mathcal{H},$$

where $\mathcal{H}$ is the Hilbert space of functions of interest and $\langle \cdot, \cdot \rangle$ denotes the inner product. This property can be understood as an approximate Plancherel formula, guaranteeing *completeness* of the set of building blocks in the function space. That is, any signal $f \in \mathcal{H}$ can be represented as an absolutely convergent infinite series of the $g_{m,n}$, or in the finite case, a linear combination thereof. Such a representation is given by the following formula:

$$f = \sum_{m,n \in \mathbb{Z}} \langle f, \tilde{g}_{m,n} \rangle g_{m,n}, \tag{1}$$

where $\tilde{g}_{m,n}$ is a dual frame for $g_{m,n}$. Dual frames exist for any frame; however, the *canonical* dual frame, guaranteeing minimal (two-)norm coefficients in the expansion of (1), is given by $\tilde{g}_{m,n} = S^{-1}g_{m,n}$, where $S$ is the *frame operator*, defined by $Sf = \sum_{m,n \in \mathbb{Z}} \langle f, g_{m,n} \rangle g_{m,n}$.

The notion of a frame thus incorporates bases as well as certain redundant representations; for example, an orthonormal basis is a *tight* frame ($A = B$) with $A = B = 1$; the union of two orthonormal bases yields a tight frame with frame bounds $A = B = 2$. Importantly, a key result in time-frequency theory (the Balian-Low Theorem) implies that redundancy is a necessary consequence of good time-frequency localisation.[1] However, even with redundancy, the frame operator may, in certain special cases, be diagonalised. If, furthermore, the $g_{m,n}$ are normalised in such a case, then analysis and synthesis can take place using the same window and inversion of the frame operator is avoided completely. Accordingly, Daubechies *et al.* [2] term such cases 'painless nonorthogonal expansions'.

## 1.2 Short-Time Spectral Attenuation

The standard noise reduction method in engineering applications is actually such an expansion in disguise (see, e.g., [3]). In this method, known as short-time spectral attenuation, a time-varying filter is applied to the frequency-domain transform of a noisy signal, using the overlap-add method of short-time Fourier analysis and synthesis. The observed signal **y** is first divided into overlapping segments through multiplication by a smooth, 'sliding' window function, which is non-zero only for a duration on the order of tens of milliseconds. The Fourier transform is then taken on each length-$l$ interval (possibly zero-padded to length $M$), and the resultant $N$ vectors of spectral values $(\mathbf{Y}_n \in \mathbb{C}^M)_{n \in 0,1,...,N-1}$ can be plotted side by side to yield a time-frequency representation known as the Gabor transform, or sub-sampled short-time Fourier transform, the modulus of which is the well-known spectrogram. The coefficients of this transform are attenuated to some degree in order to reduce the noise; as shown in Fig. 1, individual short-time intervals $\mathbf{Y}_n$ are then inverse-transformed, multiplied by a smoothing window, and added together in an appropriate manner to form a time-domain signal reconstruction $\hat{\mathbf{x}}$.

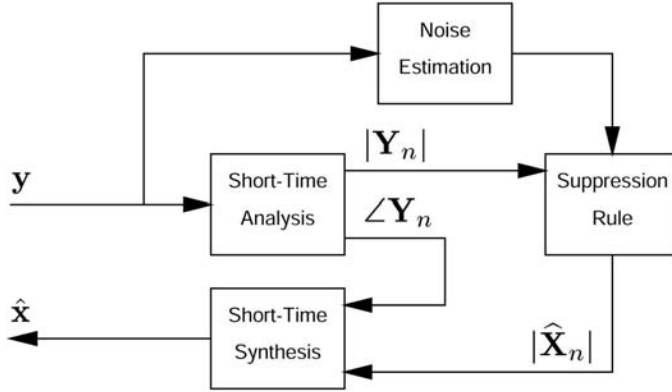

Figure 1: Short-time spectral attenuation

This method of noise reduction, while being relatively fast and easily understood, exhibits several shortcomings: in its most basic form it ignores dependencies between the time-domain data in adjacent short-time blocks, and it assumes knowledge of the noise variance. Moreover, previous approaches in this vein have relied (either explicitly or implicitly) on independence assumptions amongst the time-frequency coefficients; see, e.g., [4]. Thus, with the aim of improving upon this popular class of audio noise reduction techniques, we have used these approaches as a starting point from which to proceed with a fully Bayesian analysis. As a step in this direction, we propose a *Gabor regression model* as follows.

## 2 Coefficient Shrinkage for Audio Signal Enhancement

### 2.1 Gabor Regression

Let $\mathbf{x} \in \mathbb{R}^L$ denote a sampled audio waveform, the observation of which has been corrupted by additive white Gaussian noise of variance $\sigma^2$, yielding the simple additive model $\mathbf{y} = \mathbf{x} + \mathbf{d}$. We consider regression in this case using a design matrix obtained from a Gabor frame.[2]

In our particular case, this choice of regressors is motivated by a desire for constant absolute bandwidth, as opposed to, e.g., the constant *relative* bandwidth of wavelets. We do not attempt to address here the relative merits of Gabor and wavelet frames *per se*; rather, we simply note that the changing frequency content of natural sound signals carries much of their information, and thus a time-frequency representation may well be more appropriate than a time-scale one. Moreover, audio signal enhancement results with wavelets have been for the most part disappointing (witness the dearth of literature in this area), whereas standard engineering practice has evolved to use time-varying filtering—which is inherently Gabor analysis.

Although space does not permit a discussion of the relevance of Gabor-type transforms to auditory perception (see, e.g., [5]), as a final consideration it is interesting to note that Gabor's original formulations [6]–[7] were motivated by psychoacoustic as well as information theoretic considerations.

## 2.2 Bayesian Model

By the completeness property of Gabor frames, any $\mathbf{x} \in \mathbb{R}^L$ can be represented as a linear combination of the elements of the frame. Thus, one has the model

$$\mathbf{y} = \mathbf{Gc} + \mathbf{d},$$

where the columns of $\mathbf{G} \in \mathbb{C}^{L \times K}$ form the Gabor synthesis atoms, and elements of $\mathbf{c} \in \mathbb{C}^K$ represent the respective synthesis coefficients. To complete this model we assume an independent, identically distributed Gaussian noise vector, conditionally Gaussian coefficients, and inverted-Gamma conjugate priors:

$$\mathbf{d} \,|\, \sigma^2 \sim \mathcal{N}(0, \sigma^2 \mathbf{I}); \quad \sigma^2 \sim \mathcal{IG}(\frac{\alpha}{2}, \frac{\beta}{2})$$

$$\mathbf{c} \,|\, \boldsymbol{\sigma}_{\mathbf{c}}^2 \sim \mathcal{N}(0, \mathrm{diag}(\boldsymbol{\sigma}_{\mathbf{c}}^2)); \quad \sigma_{c_k}^2 \sim \mathcal{IG}(\kappa_k, \nu_k), \tag{2}$$

where $\mathrm{diag}(\boldsymbol{\sigma}_{\mathbf{c}}^2)$ denotes a diagonal matrix, the individual elements of which are assumed to be distributed as in (2) above, and $\alpha, \beta, \boldsymbol{\kappa}$, and $\boldsymbol{\nu}$ are hyperparameters. We note that it is possible to obtain vague priors through the choice of these hyperparameters; alternatively, one may wish to incorporate genuine prior knowledge about audio signal behaviour through them. In § 3.2, we consider the case in which frequency-dependent coefficient priors are specified in order to exploit the time-frequency structure of natural sound signals.

The choice of an inverted-Gamma prior for $\sigma^2$ is justified by its flexibility; for instance, in many audio enhancement applications one may be able to obtain a good estimate of the noise variance, which may in turn be reflected in the choice of hyperparameters $\alpha$ and $\beta$. However, in order to demonstrate the performance of our model in the 'worst-case' scenario of little prior information, we assume here a diffuse prior $(\alpha, \beta << 1)$ for $\sigma^2$.

## 2.3 Implementation

As a means of obtaining samples from the posterior distribution and hence the corresponding point estimates, we propose to sample from the posterior using Markov chain Monte Carlo (MCMC) methods [8]. By design, all model parameters may be easily sampled from their respective full conditional distributions, thus allowing the straightforward employment of a Gibbs sampler [9].

In all of the experiments described herein, a tight, normalised Hanning window was employed as the Gabor window function, and a regular time-frequency lattice was constructed to yield a redundancy of two (corresponding to the common practice of a 50% window overlap in the overlap-add method.) The arithmetic mean of the signal reconstructions from 1000 iterations (following 1000 iterations of 'burn-in', by which time the sampler appeared to have reached a stationary regime in each case) was taken to be the final result. As a further note, colour plots and representative audio examples may be found at the URL specified on the title page of this paper.

While here we show results from random initialisations, with no attempt made to optimise parameters, we note that in practice it may be most efficient to initialise the sampler with the Gabor expansion of the noisy observation vector (such an initialisation will indeed be possible without inversion of the frame operator in the cases we consider here, which correspond to the overlap-add method described in § 1.2). It can also be expected that, where possible, convergence may be speeded by starting the sampler in regions of likely high posterior probability, via use of a preliminary noise reduction method to obtain a robust coefficient initialisation.

# 3 Simulations

## 3.1 Coefficient Shrinkage in the Overcomplete Case

To test the noise reduction capabilities of the Gabor regression model, a speech signal of the short utterance 'sound check', sampled at 11.025 kHz, was artificially degraded with white Gaussian noise to yield signal-to-noise ratios (SNR) between 0 and 20 dB. At each SNR, ten runs of the sampler, at different random initialisations and using different pseudo-random number sequences, were performed as specified above. By way of comparison, three standard methods of short-time spectral attenuation (the Wiener filter, magnitude spectral subtraction, and the Ephraim and Malah suppression rule (EMSR) [4]) were also tested on the same data (noise variances were estimated from 5 seconds of the noise realisation in these cases); the results are shown in Fig. 2, along with estimates of the noise variance averaged over each of the ten runs.

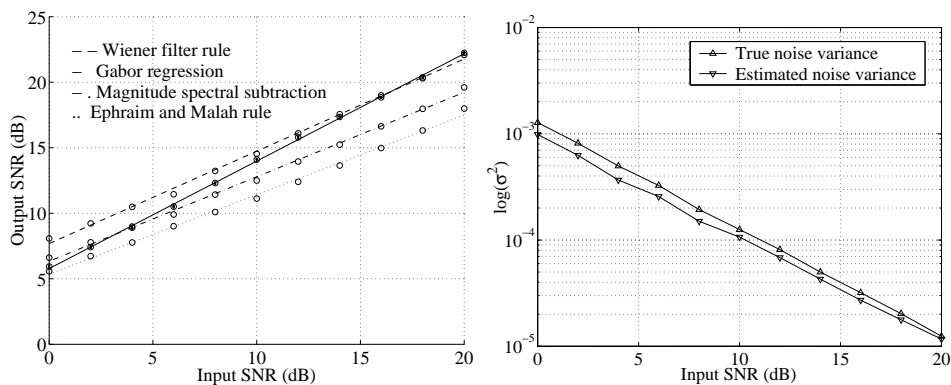

(a) Gains and corresponding interpolants. Individual realisations corresponding to the ten sampler runs are so closely spaced as to be indistinguishable.

(b) True and estimated noise variances (each averaged over ten runs of the sampler)

Figure 2: Noise reduction results for the Gabor regression experiment of § 3.1

As it is able to outperform many of the short-time methods over a wide range of SNR (despite its relative disadvantage of not being given the estimated noise variance), and is also able to accurately estimate the noise variance over this range, the results of Fig. 2 would seem to indicate the appropriateness of the Gabor regression scheme for audio signal enhancement. However, listening tests reveal that the algorithm, while improving upon the shortcomings of standard approaches discussed in § 1.2, still suffers from the same 'musical' residual noise. The EMSR, on the other hand, is known for its more colourless residual noise (although as can be seen from Fig. 2, it tends to exhibit severe over-smoothing at higher SNR); we address this issue in the following section.

## 3.2 Coefficient Shrinkage Using Wilson Bases

In the case of a real signal, it is still possible to obtain good time-frequency localisation without incurring the penalty of redundancy through the use of *Wilson bases* (also known in the engineering literature as lapped transforms; see, e.g., [1]).

As an example of incorporating basic prior knowledge about audio signal structure in a relatively simple and straightforward manner, now consider letting the scale factor $\nu$ of (2) become an inverse function of frequency, so that elements of the inverted-Gamma-distributed coefficient variance vector $\sigma_c^2$, although independent, are no longer identically distributed.

To test the effects of such a frequency-dependent prior in the context of a Wilson regression model (in comparison with the diffuse priors employed in § 3.1), the speech signal of the previous example was degraded with white Gaussian noise of variance $1.26 \times 10^{-4}$, to yield an SNR of 10 dB. Once again, posterior mean estimates over the last 1000 iterations of a 2000-iteration Gibbs sampler run were taken as the final result. Figure 3 shows samples of the noise variance parameter in this case. While both the diffuse and frequency-dependent

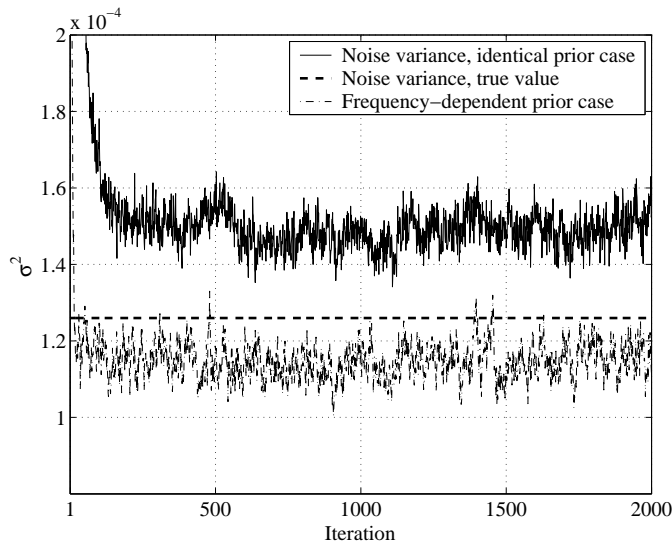

Figure 3: Noise variance samples for the two Wilson regression schemes of § 3.2

prior schemes yield an estimate close to the true noise variance, and indeed give similar SNR gains of 3.07 and 2.85 dB, respectively, the corresponding restorations differ greatly in their perceptual quality. Figure 4 shows spectrograms of the clean and noisy test signal, as well as the resultant restorations; whereas Fig. 5 shows waveform and spectrogram plots of the corresponding residuals (for greater clarity, colour plots are provided on-line).

It may be seen from Figs. 4 and 5 that the residual noise in the case of the frequency-dependent priors appears less coloured, and in fact this restoration suffers much less from the so-called 'musical noise' artefact common to audio signal enhancement methods. It is well-known that a 'whiter-sounding' residual is perceptually preferable; in fact, some noise reduction methods have attempted this explicitly [10].

## 4   Discussion

Here we have presented a model for regression of audio signals, using elements of a Gabor frame as a design matrix. Note that in alternative contexts, others have also considered scale mixtures of normals as we do here (see, e.g., [11]–[12]); in fact, the priors discussed in [13] constitute special cases of those employed in the Gabor regression model. This model may also be extended to include indicator variables, thus allowing one to perform Bayesian model averaging [8]–[9]. In this case it may be desirable to employ an even larger

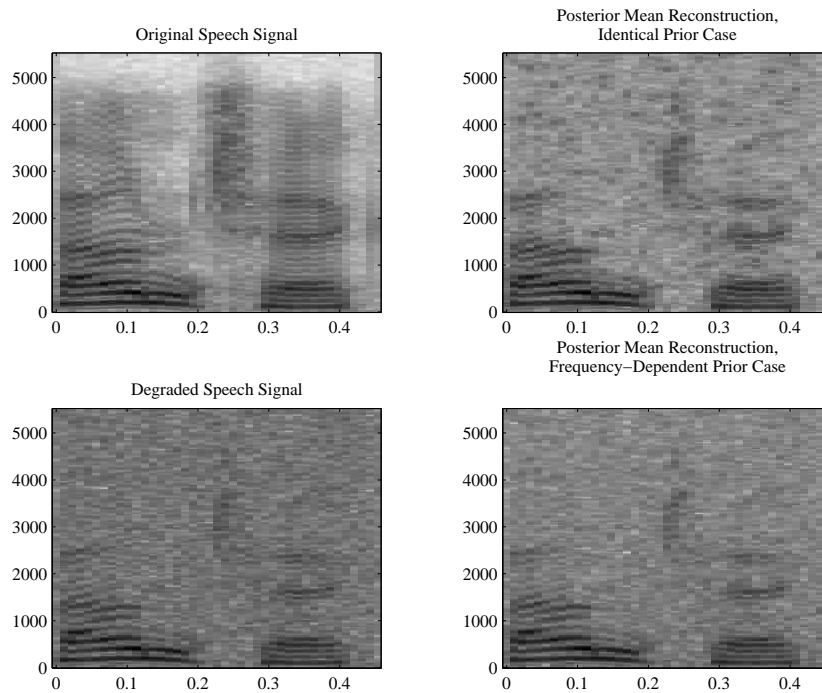

Figure 4: Spectrograms for the two Wilson regression schemes of § 3.2 in the case of diffuse *vs.* frequency-dependent priors (grey scale is proportional to log-amplitude)

'dictionary' of regressors, in order to obtain the most parsimonious representation possible.[3] Multi-resolution wavelet-like schemes are one of many possibilities; for an example application in this vein we refer the reader to [14].

The strength of such a fully Bayesian approach lies largely in its extensibility to allow for more accurate signal and noise models; in this vein work is continuing on the development of appropriate conditional prior structures for audio signals, including the formulation of Markov random field models. The main weakness of this method at present lies in the computational intensity inherent in the sampling scheme; a comparison to more recent and sophisticated probabilistic methods (e.g., [15]–[16]) is now in order to determine whether the benefits to be gained from such an approach outweigh its computational drawbacks.

## Footnotes

[1]There is, however, an exception for real signals, which will be explored in more detail in § 3.2.

[2]Technically, we consider the ring $\mathbb{Z}_L = \mathbb{Z}$ mod $L$, under the assumption (without loss of generality) that the vector of sampled observations $\mathbf{y}$ has been extended to length $L$ in a proper way at its boundary before being periodically extended on $\mathbb{Z}_L$.

[3]It remains an open question as to whether the resultant variable selection problem would be amenable to approaches other than MCMC—for instance, a perfect sampling scheme.

## References

[1] Gröchenig, K. (2001). *Foundations of Time-Frequency Analysis*. Boston: Birkhäuser.

[2] Daubechies, I., Grossmann, A., and Meyer, Y. (1986). Painless nonorthogonal expansions. *J. Math. Phys.* **27**, 1271–1283.

[3] Dörfler, M. (2001). Time-frequency analysis for music signals: a mathematical approach. *J. New Mus. Res.* **30**, 3–12.

[4] Ephraim, Y. and Malah, D. (1984). Speech enhancement using a minimum mean-square error short-time spectral amplitude estimator. *IEEE Trans. Acoust., Speech, Signal Processing* **ASSP-32**, 1109–1121.

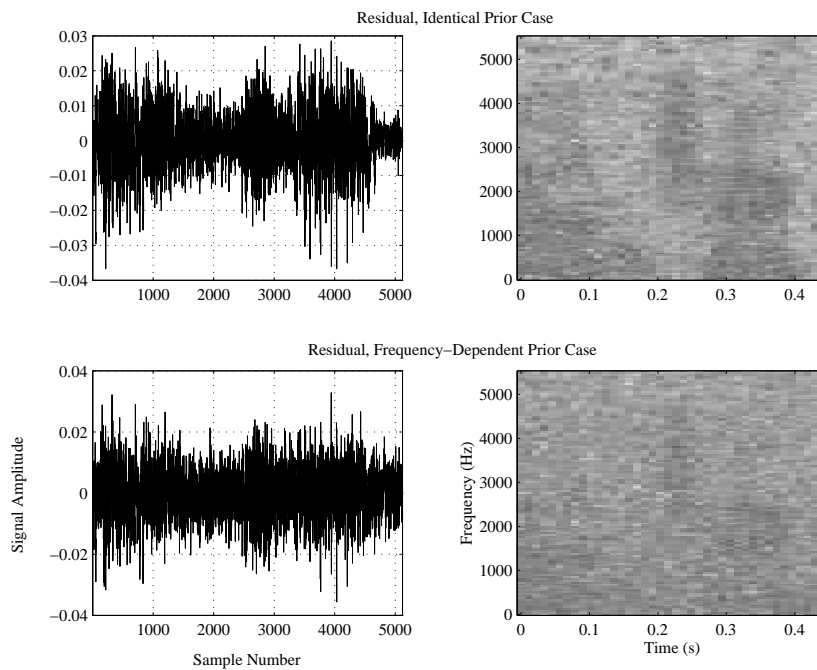

Figure 5: Waveform and spectrogram plots of the Wilson regression residuals

[5] Wolfe, P. J. and Godsill, S. J. (2001). Perceptually motivated approaches to music restoration. *J. New Mus. Res.* **30**, 83–92.

[6] Gabor, D. (1946). Theory of communication. *J. IEE* **93**, 429–457.

[7] Gabor, D. (1947). Acoustical quanta and the theory of hearing. *Nature* **159**, 591–594.

[8] Robert, C. P. and Casella, G. (1999). *Monte Carlo Statistical Methods.* New York: Springer.

[9] Gilks, W. R., Richardson, S., and Spiegelhalter, D. J. (1996). *Markov Chain Monte Carlo in Practice.* London: Chapman & Hall.

[10] Ephraim, Y. and Van Trees, H. L. (1995). A signal subspace approach for speech enhancement. *IEEE Trans. Speech Audio Processing* **3**, 251–266.

[11] Shepard, N. (1994). Partial non-Gaussian state space. *Biometrika* **81**, 115–131.

[12] Godsill, S. J. and Rayner, P. J. W. (1998). *Digital Audio Restoration: A Statistical Model Based Approach.* Berlin: Springer-Verlag.

[13] Figueiredo, M. A. T. (2002). Adaptive sparseness using Jeffreys prior. In T. G. Dietterich, S. Becker, and Z. Ghahramani (eds.), *Advances in Neural Information Processing Systems 14*, pp. 697–704. Cambridge, MA: MIT Press.

[14] Wolfe, P. J., Dörfler, M., and Godsill, S. J. (2001). Multi-Gabor dictionaries for audio time-frequency analysis. In *Proc. IEEE Worksh. App. Signal Processing Audio Acoust.*, pp. 43–46.

[15] H. Attias, L. Deng, A. Acero, and J. C. Platt (2001). A new method for speech denoising and robust speech recognition using probabilistic models for clean speech and for noise. In *Proc. Eurospeech 2001*, vol. 3, pp. 1903–1906.

[16] H. Attias, J.C. Platt, A. Acero, and L. Deng (2001). Speech denoising and dereverberation using probabilistic models. In T. Leen (ed.), *Advances in Neural Information Processing Systems 13*, pp. 758–764. Cambridge, MA: MIT Press.
